# Transductive and Inductive Methods for Approximate Gaussian Process Regression

**Anton Schwaighofer**[1,2]
[1] TU Graz, Institute for Theoretical Computer Science
Inffeldgasse 16b, 8010 Graz, Austria
`http://www.igi.tugraz.at/aschwaig`

**Volker Tresp**[2]
[2] Siemens Corporate Technology CT IC4
Otto-Hahn-Ring 6, 81739 Munich, Germany
`http://www.tresp.org`

## Abstract

Gaussian process regression allows a simple analytical treatment of exact Bayesian inference and has been found to provide good performance, yet scales badly with the number of training data. In this paper we compare several approaches towards scaling Gaussian processes regression to large data sets: the subset of representers method, the reduced rank approximation, online Gaussian processes, and the Bayesian committee machine. Furthermore we provide theoretical insight into some of our experimental results. We found that subset of representers methods can give good and particularly fast predictions for data sets with high and medium noise levels. On complex low noise data sets, the Bayesian committee machine achieves significantly better accuracy, yet at a higher computational cost.

## 1 Introduction

Gaussian process regression (GPR) has demonstrated excellent performance in a number of applications. One unpleasant aspect of GPR is its scaling behavior with the size of the training data set $N$. In direct implementations, training time increases as $O(N^3)$, with a memory footprint of $O(N^2)$. The subset of representer method (SRM), the reduced rank approximation (RRA), online Gaussian processes (OGP) and the Bayesian committee machine (BCM) are approaches to solving the scaling problems based on a finite dimensional approximation to the typically infinite dimensional Gaussian process.

The focus of this paper is on providing a unifying view on the methods and analyze their differences, both from an experimental and a theoretical point of view. For all of the discussed methods, we also examine asymptotic and actual runtime and investigate the accuracy versus speed trade-off. A major difference of the methods discussed here is that the BCM performs *transductive* learning, whereas RRA, SRM and OGP methods perform

*induction* style learning. By transduction[1] we mean that a particular method computes a test set dependent model, i.e. it exploits knowledge about the *location of the test data* in its approximation. As a consequence, the BCM approximation is calculated when the inputs to the test data are known. In contrast, inductive methods (RRA, OGP, SRM) build a model solely on basis of information from the training data.

In Sec. 1.1 we will briefly introduce Gaussian process regression (GPR). Sec. 2 presents the various inductive approaches to scaling GPR to large data, Sec. 3 follows with transductive approaches. In Sec. 4 we give an experimental comparison of all methods and an analysis of the results. Conclusions are given in Sec. 5.

### 1.1 Gaussian Process Regression

We consider Gaussian process regression (GPR) on a set of training data $\mathcal{D} = \{(\mathbf{x}_i, y_i)\}_{i=1}^{N}$, where targets are generated from an unknown function $f$ via $y_i = f(\mathbf{x}_i) + e_i$ with independent Gaussian noise $e_i$ of variance $\sigma^2$. We assume a Gaussian process prior on $f(\mathbf{x}_i)$, meaning that functional values $f(\mathbf{x}_i)$ on points $\{\mathbf{x}_i\}_{i=1}^{N}$ are jointly Gaussian distributed, with zero mean and covariance matrix (or Gram matrix) $K^N$. $K^N$ itself is given by the kernel (or covariance) function $k(\cdot, \cdot)$, with $K_{ij}^N = k(\mathbf{x}_i, \mathbf{x}_j)$.

The Bayes optimal estimator $\hat{f}(x) = E(f(x)|\mathcal{D})$ takes on the form of a weighted combination of kernel functions [4] on training points $\mathbf{x}_i$

$$\hat{f}(\mathbf{x}) = \sum_{i=1}^{N} w_i k(\mathbf{x}, \mathbf{x}_i). \tag{1}$$

The weight vector $\mathbf{w} = (w_1, \ldots, w_N)^\top$ is the solution to the system of linear equations

$$(K^N + \sigma^2 \mathbf{1})\mathbf{w} = \mathbf{y} \tag{2}$$

where $\mathbf{1}$ denotes a unit matrix and $\mathbf{y} = (y_1, \ldots, y_N)^\top$. Mean and covariance of the GP prediction $\mathbf{f}^*$ on a set of test points $\mathbf{x}_1^*, \ldots, \mathbf{x}_T^*$ can be written conveniently as

$$E(\mathbf{f}^*|\mathcal{D}) = K^{*N}\mathbf{w} \text{ and } \operatorname{cov}(\mathbf{f}^*|\mathcal{D}) = K^* - K^{*N}(K^N + \sigma^2 \mathbf{1})^{-1}(K^{*N})^\top \tag{3}$$

with $K_{ij}^{*N} = k(\mathbf{x}_i^*, \mathbf{x}_j)$. Eq. (2) shows clearly what problem we may expect with large training data sets: The solution to a system of $N$ linear equations requires $O(N^3)$ operations, and the size of the Gram matrix $K^N$ may easily exceed the memory capacity of an average work station.

## 2 Inductive Methods for Approximate GPR

### 2.1 Reduced Rank Approximation (RRA)

Reduced rank approximations focus on ways of efficiently solving the system of linear equations Eq. (2), by replacing the kernel matrix $K^N$ with some approximation $\tilde{K}^N$.

Williams and Seeger [12] use the Nyström method to calculate an approximation to the first $B$ eigenvalues and eigenvectors of $K^N$. Essentially, the Nyström method performs an eigendecomposition of the $B \times B$ covariance matrix $K^B$, obtained from a set of $B$ basis points selected at random out of the training data. Based on the eigendecomposition of $K^B$,

one can compute approximate eigenvalues and eigenvectors of $K^N$. In a special case, this reduces to

$$K^N \approx \tilde{K}^N = K^{NB}(K^B)^{-1}(K^{NB})^\top. \tag{4}$$

where $K^B$ is the kernel matrix for the set of basis points, and $K^{NB}$ is the matrix of kernel evaluations between training and basis points. Subsequently, this can be used to obtain an approximate solution $\tilde{\mathbf{w}}$ of Eq. (1) via matrix inversion lemma in $O(NB^2)$ instead of $O(N^3)$.

## 2.2 Subset of Representers Method (SRM)

Subset of representers methods replace Eq. (1) by a linear combination of kernel functions on a set of $B$ basis points, leading to an approximate predictor

$$\tilde{f}(\mathbf{x}) = \sum_{i=1}^{B} \beta_i k(\mathbf{x}, \mathbf{x}_i) \tag{5}$$

with an optimal weight vector

$$\beta = (\sigma^2 K^B + (K^{NB})^\top K^{NB})^{-1}(K^{NB})^\top \mathbf{y}. \tag{6}$$

Note that Eq. (5) becomes exact if the kernel function allows a decomposition of the form $k(\mathbf{x}_i, \mathbf{x}_j) = K^{i,B}(K^B)^{-1}(K^{j,B})^\top$.

In practical implementation, one may expect different performance depending on the choice of the $B$ basis points $\mathbf{x}_1, \ldots, \mathbf{x}_B$. Different approaches for basis selection have been used in literature, we will discuss them in turn.

Obviously, one may select the basis points at random (SRM Random) out of the training set. While this produces no computational overhead, the prediction outcome may be sub-optimal.

In the sparse greedy matrix approximation (SRM SGMA, [6]) a subset of $B$ basis kernel functions is selected such that all kernel functions on the training data can be well approximated by linear combinations of the selected basis kernels[2]. If proximity in the associated reproducing kernel Hilbert space (RKHS) is chosen as the approximation criterion, the optimal linear combination (for a given basis set) can be computed analytically. Smola and Schölkopf [6] introduce a greedy algorithm that finds a near optimal set of basis functions, where the algorithm has the same asymptotic complexity $O(NB^2)$ as the SRM Random method.

Whereas the SGMA basis selection focuses only on the representation power of kernel functions, one can also design a basis selection scheme that takes into account the full likelihood model of the Gaussian process. The underlying idea of the greedy posterior approximation algorithm (SRM PostApp, [7]) is to compare the log posterior of the subset of representers method and the full Gaussian process log posterior. One thus can select basis functions in such a fashion that the SRM log posterior best approximates[3] the full GP log posterior, while keeping the total number of basis functions $B$ minimal. As for the case of SGMA, this algorithm can be formulated such that its asymptotic computational complexity is $O(NB^2)$, where $B$ is the total number of basis functions selected.

## 2.3 Online Gaussian Processes

Csató and Opper [2] present an online learning scheme that focuses on a sparse model of the posterior process that arises from combining a Gaussian process prior with a general

likelihood model of data. The posterior process is assumed to be Gaussian and is modeled by a set of basis vectors. Upon arrival of a new data point, the updated (possibly non-Gaussian) posterior process is being projected to the closest (in a KL-divergence sense) Gaussian posterior. If this projection induces an error above a certain threshold, the newly arrived data point will be included in the set of basis vectors. Similarly, basis vectors with minimum contribution to the posterior process may be removed from the basis set.

## 3  Transductive Methods for Approximate GPR

In order to derive a transductive kernel classifier, we rewrite the Bayes optimal prediction Eq. (3) as follows:

$$E(\mathbf{f}^*|\mathcal{D}) = K^* \left[ K^* + K^{*N} \text{cov}(\mathbf{y}|\mathbf{f}^*)^{-1} (K^{*N})^\top \right]^{-1} K^{*N} \text{cov}(\mathbf{y}|\mathbf{f}^*)^{-1} \mathbf{y}. \tag{7}$$

Here, $\text{cov}(\mathbf{y}|\mathbf{f}^*)$ is the covariance obtained when predicting training observations $\mathbf{y}$ given the functional values $\mathbf{f}^*$ at the test points:

$$\text{cov}(\mathbf{y}|\mathbf{f}^*) = K^N + \sigma^2 \mathbf{1} - (K^{*N})^\top (K^*)^{-1} K^{*N} \tag{8}$$

Mind that this matrix can be written down without actual knowledge of $\mathbf{f}^*$.

Examining Eq. (7) reveals that the Bayes optimal prediction of Eq. (3) can be expressed as a weighted sum of kernel functions on test points. In Eq. (7), the term $\text{cov}(\mathbf{y}|\mathbf{f}^*)^{-1}\mathbf{y}$ gives a weighting of training observations $\mathbf{y}$: Training points which cannot be predicted well from the functional values of the test points are given a lower weight. Data points which are "closer" to the test points (in the sense that they can be predicted better) obtain a higher weight than data which are remote from the test points.

Eq. (7) still involves the inversion of the $N \times N$ matrix $\text{cov}(\mathbf{y}|\mathbf{f}^*)^{-1}$ and thus does not make a practical method. By using different approximations for $\text{cov}(\mathbf{y}|\mathbf{f}^*)^{-1}$, we obtain different transductive methods, which we shall discuss in the next sections.

Note that in a Bayesian framework, transductive and inductive methods are equivalent, if we consider matching models (the true model for the data is in the family of models we consider for learning). Large data sets reveal more of the structure of the true model, but for computational reasons, we may have to limit ourselves to models with lower complexity. In this case, transductive methods allow us to focus on the actual region of interest, i.e. we can build models that are particularly accurate in the region where the test data lies.

### 3.1  Transductive SRM

For large sets of test data, we may assume $\text{cov}(\mathbf{y}|\mathbf{f}^*)$ to be a diagonal matrix $\text{cov}(\mathbf{y}|\mathbf{f}^*) \approx \sigma^2 \mathbf{1}$, meaning that test values $\mathbf{f}^*$ allow a perfect prediction of training observations (up to noise). With this approximation, Eq. (7) reduces to the prediction of a subset of representers method (see Sec. 2.2) where the test points are used as the set of basis points (SRM Trans).

### 3.2  Bayesian Committee Machine (BCM)

For a smaller number of test data, assuming a diagonal matrix for $\text{cov}(\mathbf{y}|\mathbf{f}^*)$ (as for the transductive SRM method) seems unreasonable. Instead, we can use the less stringent assumption of $\text{cov}(\mathbf{y}|\mathbf{f}^*)$ being block diagonal. After some matrix manipulations, we obtain

the following approximation for Eq. (7) with block diagonal $\mathrm{cov}(\mathbf{y}|\mathbf{f}^*)$:

$$\hat{E}(\mathbf{f}^*|\mathcal{D}) = C^{-1} \sum_{i=1}^{M} \mathrm{cov}(\mathbf{f}^*|\mathcal{D}^i)^{-1} E(\mathbf{f}^*|\mathcal{D}^i) \qquad (9)$$

$$C = \widehat{\mathrm{cov}}(\mathbf{f}^*|\mathcal{D})^{-1} = -(M-1)(K^*)^{-1} + \sum_{i=1}^{M} \mathrm{cov}(\mathbf{f}^*|\mathcal{D}^i)^{-1}. \qquad (10)$$

This is equivalent to the Bayesian committee machine (BCM) approach [8]. In the BCM, the training data $\mathcal{D}$ are partitioned into $M$ disjoint sets $\mathcal{D}^1, \ldots, \mathcal{D}^M$ of approximately same size ("modules"), and $M$ GPR predictors are trained on these subsets. In the prediction stage, the BCM calculates the unknown responses $\mathbf{f}^*$ at a set of test points $\mathbf{x}_1^* \ldots \mathbf{x}_T^*$ at once. The prediction $E(\mathbf{f}^*|\mathcal{D}^i)$ of GPR module $i$ is weighted by the inverse covariance of its prediction. An intuitively appealing effect of this weighting scheme is that modules which are uncertain about their predictions are automatically weighted less than modules that are certain about their predictions.

Very good results were obtained with the BCM with random partitioning [8] into subsets $\mathcal{D}^i$. The block diagonal approximation of $\mathrm{cov}(\mathbf{y}|\mathbf{f}^*)$ becomes particularly accurate, if each $\mathcal{D}^i$ contains data that is spatially separated from other training data. This can be achieved by pre-processing the training data with a simple $k$-means clustering algorithm, resulting in an often drastic reduction of the BCM's error rates. In this article, we always use the BCM with clustered data.

## 4   Experimental Comparison

In this section we will present an evaluation of the different approximation methods discussed in Sec. 2 and 3 on four data sets. In the ABALONE data set [1] with 4177 examples, the goal is to predict the age of Abalones based on 8 inputs. The KIN8NM data set[4] represents the forward dynamics of an 8 link all-revolute robot arm, based on 8192 examples. The goal is to predict the distance of the end-effector from a target, given the twist angles of the 8 links as features. KIN40K represents the same task, yet has a lower noise level than KIN8NM and contains 40.000 examples. Data set ART with 50000 examples was used extensively in [8] and describes a nonlinear map with 5 inputs with a small amount of additive Gaussian noise.

For all data sets, we used a squared exponential kernel of the form $k(\mathbf{x}_i, \mathbf{x}_j) = \exp\left(-\frac{1}{2d^2}\|\mathbf{x}_i - \mathbf{x}_j\|^2\right)$, where the kernel parameter $d$ was optimized individually for each method. To allow a fair comparison, the subset selection methods SRM SGMA and SRM PostApp were forced to select a given number $B$ of basis functions (instead of using the stopping criteria proposed by the authors of the respective methods). Thus, all methods form their predictions as a linear combination of exactly $B$ basis functions.

Table 1 shows the average remaining variance[5] in a 10-fold cross validation procedure on all data sets. For each of the methods, we have run experiments with different kernel width $d$. In Table 1 we list only the results obtained with optimal $d$ for each method.

On the ABALONE data set (very high level of noise), all of the tested methods achieved almost identical performance, both with $B = 200$ and $B = 1000$ basis functions. For all other data sets, significant performance differences were observed. Out of the inductive

| Method | Abalone 200 | Abalone 1000 | KIN8NM 200 | KIN8NM 1000 | KIN40K 200 | KIN40K 1000 | ART 200 | ART 1000 |
|---|---|---|---|---|---|---|---|---|
| SRM PostApp | 42.81 | 42.81 | 13.79 | **7.84** | 9.49 | 2.36 | 3.91 | 1.12 |
| SRM SGMA | 42.83 | 42.81 | 21.84 | 8.70 | 18.32 | 4.25 | 5.62 | 1.79 |
| SRM Random | 42.86 | 42.82 | 22.34 | 9.01 | 18.77 | 4.39 | 5.87 | 1.79 |
| RRA Nyström | 42.98 | 41.10 | N/A | N/A | N/A | N/A | N/A | N/A |
| Online GP | 42.87 | N/A | 16.49 | N/A | 10.36 | N/A | 5.37 | N/A |
| BCM | 42.86 | 42.81 | **10.32** | 8.31 | **2.81** | **0.83** | **0.27** | **0.20** |
| SRM Trans | 42.93 | 42.79 | 21.95 | 9.79 | 16.47 | 4.25 | 5.15 | 1.64 |

Table 1: Remaining variance, obtained with different GPR approximation methods on four data sets, with different number of basis functions selected (200 or 1000). Remaining variance is given in per cent, averaged over 10-fold cross validation. Marked in bold are results that are significantly better (with a significance level of 99% or above in a paired $t$-test) than any of the other methods

methods (SRM SGMA, SRM Random, SRM PostApp, RRA Nyström) best performance was always achieved with SRM PostApp. Using the results in a paired $t$-test showed that this was significant at a level of 99% or above. Online Gaussian processes[6] typically performed slightly worse than SRM PostApp. Furthermore, we observed certain problems with the RRA Nyström method. On all but the ABALONE data set, weights $\tilde{\mathbf{w}}$ took on values in the range of $10^3$ or above, leading to poor performance. For this reason, the results for RRA Nyström were omitted from Table 1. Further comments on these problems will be given in Sec. 4.2.

Comparing induction and transduction methods, we see that the BCM performs significantly better than any inductive method in most cases. Here, the average MSE obtained with the BCM was only a fraction (25-30%) of the average MSE of the best inductive method. By a paired $t$-test we confirmed that the BCM is significantly better than all other methods on the KIN40K and ART data sets, with significance level of 99% or above. On the KIN8NM data set (medium noise level) we observed a case where SRM PostApp performed best. We attribute this to the fact that k-means clustering was not able to find well separated clusters. This reduces the performance of the BCM, since the block diagonal approximation of Eq. (8) becomes less accurate (see Sec. 3.2). Mind that all transductive methods necessarily lose their advantage over inductive methods, when the allowed model complexity (that is, the number of basis functions) is increased.

We further noticed that, on the KIN40K and ART data sets, SRM Trans consistently outperformed SRM Random, despite of SRM Trans being the most simplistic transductive method. The difference in performance was only small, yet significant at a level of 99%.

As mentioned above, we did not make use of the stopping criterion proposed for the SRM PostApp method, namely the relative gap between SRM log posterior and the log posterior of the full Gaussian process model. In [7], the authors suggest that the gap is indicative of the generalization performance of the SRM model and use a gap of 2.5% in their experiments. In contrast, we did not observe any correlation between the gap and the generalization performance in our experiments. For example, selecting 200 basis points out of the KIN40K data set gave a gap of $\approx 1\%$, indicating a good fit. As shown in Table 1, a significantly better error was achieved with 1000 basis functions (giving a gap of $\approx 3.5 \cdot 10^{-4}$). Thus, it remains open how one can automatically choose an appropriate basis set size $B$.

| Method | Memory consumption | | Computational cost | | Runtime |
|---|---|---|---|---|---|
| | Initialization | Prediction | Initialization | Prediction | KIN40K |
| Exact GPR | $O(N^2)$ | $O(N)$ | $O(N^3)$ | $O(N)$ | N/A |
| RRA Nyström | $O(NB)$ | $O(N)$ | $O(NB^2)$ | $O(N)$ | 4 min |
| SRM Random | $\left.\begin{array}{c}\\\\\\\\\end{array}\right\}O(NB)$ | $\left.\begin{array}{c}\\\\\\\\\end{array}\right\}O(B)$ | $\left.\begin{array}{c}\\\\\\\\\end{array}\right\}O(NB^2)$ | $\left.\begin{array}{c}\\\\\\\\\end{array}\right\}O(B)$ | 3 min |
| SRM Trans | | | | | 3 min |
| SRM SGMA | | | | | 7 h |
| SRM PostApp | | | | | 11 h |
| Online GP | $O(B^2)$ | $O(B)$ | $O(NB^2)$ | $O(B)$ | est. 150 h |
| BCM | — | $O(N+B^2)$ | — | $O(NB)$ | 30 min |

Table 2: Memory consumption, asymptotic computational cost and actual runtime for different GP approximation methods with $N$ training data points and $B$ basis points, $B < N$. For the BCM, we assume here that training and test data are partitioned into modules of size $B$. Asymptotic cost for predictions show the cost per test point. The actual runtime is given for the KIN40K data set, with 36000 training examples, 4000 test patterns and $B = 1000$ basis functions for each method.

## 4.1 Computational Cost

Table 2 shows the asymptotic computational cost for all approximation methods we have described in Sec. 2 and 3. The subset of representers methods (SRM) show the most favorable cost for the prediction stage, since the resulting model consists only of $B$ basis functions with their associated weight vector. Table 2 also lists the actual runtime[7] for one (out of 10) cross validation runs on the KIN40K data set. Here, methods with the same asymptotic complexity exhibit runtimes ranging from 3 minutes to 150 hours. For the SRM methods, most of this time is spent for basis selection (SRM PostApp and SRM SGMA). We thus consider the slow basis selection as the bottleneck for SRM methods when working with larger number of basis functions or larger data sets.

## 4.2 Problems with RRA Nyström

As mentioned in Sec. 4, we observed that weights $\tilde{\mathbf{w}}$ in RRA Nyström take on values in the range of $10^3$ or above on data sets KIN8NM, KIN40K and ART. This can be explained by considering the perturbation of linear systems. RRA Nyström solves Eq. (2) with an approximate $\tilde{K}^N$ instead of $K^N$, thus calculating an approximate $\tilde{\mathbf{w}}$ instead of the true $\mathbf{w}$. Using matrix perturbation theory, we can show that the relative error of the approximate $\tilde{\mathbf{w}}$ is bounded by

$$\frac{\|\tilde{\mathbf{w}} - \mathbf{w}\|}{\|\mathbf{w}\|} \leq \max_i \frac{|\lambda_i - \tilde{\lambda}_i|}{\tilde{\lambda}_i + \sigma^2} \tag{11}$$

where $\lambda_i$ and $\tilde{\lambda}_i$ denote eigenvalues of $K^N$ resp. $\tilde{K}^N$. A closer look at the Nyström approximation [11] revealed that already for moderately complex data sets, such as KIN8NM, it tends to underestimate eigenvalues of the Gram matrix, unless a very high number of basis points is used. If in addition a rather low noise variance is assumed, we obtain a very high value for the error bound in Eq. (11), confirming our observations in the experiments. Methods to overcome the problems associated with the Nyström approximation are currently being investigated [11].

# 5 Conclusions

Our results indicate that, depending on the computational resources and the desired accuracy, one may select methods as follows: If the major concern is speed of prediction, one is well advised to use the subset of representers method with basis selection by greedy posterior approximation. This method may be expected to give results that are significantly better than other (inductive) methods. While being painfully slow during basis selection, the resulting models are compact, easy to use and accurate. Online Gaussian processes achieve a slightly worse accuracy, yet they are the only (inductive) method that can easily be adapted for general likelihood models, such as classification and regression with non-Gaussian noise. A generalization of the BCM to non-Gaussian likelihood models has been presented in [9].

On the other hand, if accurate predictions are the major concern, one may expect best results with the Bayesian committee machine. On complex low noise data sets (such as KIN40K and ART) we observed significant advantages in terms of prediction accuracy, giving an average mean squared error that was only a fraction (25-30%) of the error achieved by the best inductive method. For the BCM, one must take into account that it is a transduction scheme, thus prediction time and memory consumption are larger than those of SRM methods.

Although all discussed approaches scale linearly in the number of training data, they exhibit significantly different runtime in practice. For the experiments we had done in this paper (running 10-fold cross validation on given data) the Bayesian committee machine is about one order of magnitude slower than an SRM method with randomly chosen basis; SRM with greedy posterior approximation is again an order of magnitude slower than the BCM.

**Acknowledgements**   Anton Schwaighofer gratefully acknowledges support through an Ernst-von-Siemens scholarship.

## Footnotes

[1]Originally, the differences between transductive and inductive learning where pointed out in statistical learning theory [10]. Inductive methods minimize the expected loss over all possible test sets, whereas transductive methods minimize the expected loss for one particular test set.

[2]This method was not developed particularly for GPR, yet we expect this basis selection scheme to be superior to a purely random choice.

[3]However, Rasmussen [5] noted that Smola and Bartlett [7] falsely assume that the additive constant terms in the log likelihood remain constant during basis selection.

[4]From the DELVE archive `http://www.cs.toronto.edu/~delve/`

[5]remaining variance $= 100 \times \frac{MSE_{\text{model}}}{MSE_{\text{mean}}}$, where $MSE_{\text{mean}}$ is the MSE obtained from using the mean of training targets as the prediction for all test data. This gives a measure of performance that is independent of data scaling.

[6]Due to the numerically demanding approximations, runtime of the OGP method for $B = 1000$ is rather long. We thus only list results for $B = 200$ basis functions.

[7]Runtime was logged on Linux PCs with AMD Athlon 1GHz CPUs, with all methods implemented in Matlab and optimized with the Matlab profiler.

## References

[1] Blake, C. and Merz, C. UCI repository of machine learning databases. 1998.

[2] Csató, L. and Opper, M. Sparse online gaussian processes. *Neural Computation*, 14(3):641–668, 2002.

[3] Leen, T. K., Dietterich, T. G., and Tresp, V., eds. *Advances in Neural Information Processing Systems 13*. MIT Press, 2001.

[4] MacKay, D. J. Introduction to Gaussian processes. In C. M. Bishop, ed., *Neural Networks and Machine Learning*, vol. 168 of *NATO Asi Series. Series F, Computer and Systems Sciences*. Springer Verlag, 1998.

[5] Rasmussen, C. E. Reduced rank Gaussian process learning, 2002. Unpublished Manuscript.

[6] Smola, A. and Sch¨olkopf, B. Sparse greedy matrix approximation for machine learning. In P. Langely, ed., *Proceedings of ICML00*. Morgan Kaufmann, 2000.

[7] Smola, A. J. and Bartlett, P. Sparse greedy gaussian process regression. In [3], pp. 619–625.

[8] Tresp, V. A Bayesian committee machine. *Neural Computation*, 12(11):2719–2741, 2000.

[9] Tresp, V. The generalized bayesian committee machine. In *Proceedings of the Sixth ACM SIGKDD International Conference on Knowledge Discovery and Data Mining*, pp. 130–139. Boston, MA USA, 2000.

[10] Vapnik, V. N. *The nature of statistical learning theory*. Springer Verlag, 1995.

[11] Williams, C. K., Rasmussen, C. E., Schwaighofer, A., and Tresp, V. Observations on the Nystr¨om method for Gaussian process prediction. Tech. rep., Available from the authors' web pages, 2002.

[12] Williams, C. K. I. and Seeger, M. Using the nystr¨om method to speed up kernel machines. In [3], pp. 682–688.